# Adaptive choice of grid and time in reinforcement learning

**Stephan Pareigis**
stp@numerik.uni-kiel.de
Lehrstuhl Praktische Mathematik
Christian-Albrechts-Universität Kiel
Kiel, Germany

## Abstract

We propose local error estimates together with algorithms for adaptive a-posteriori grid and time refinement in reinforcement learning. We consider a deterministic system with continuous state and time with infinite horizon discounted cost functional. For grid refinement we follow the procedure of numerical methods for the Bellman-equation. For time refinement we propose a new criterion, based on consistency estimates of discrete solutions of the Bellman-equation. We demonstrate, that an optimal ratio of time to space discretization is crucial for optimal learning rates and accuracy of the approximate optimal value function.

## 1 Introduction

Reinforcement learning can be performed for fully continuous problems by discretizing state space and time, and then performing a discrete algorithm like $Q$-learning or RTDP (e.g. [5]). Consistency problems arise if the discretization needs to be refined, e.g. for more accuracy, application of multi-grid iteration or better starting values for the iteration of the approximate optimal value function. In [7] it was shown, that for diffusion dominated problems, a state to time discretization ratio $k/h$ of $Ch^{\gamma}$, $\gamma > 0$ has to hold, to achieve consistency (i.e. $k = o(h)$). It can be shown, that for deterministic problems, this ratio must only be $k/h = C$, $C$ a constant, to get consistent approximations of the optimal value function. The choice of the constant $C$ is crucial for fast learning rates, optimal use of computer memory resources and accuracy of the approximation.

We suggest a procedure involving local a-posteriori error estimation for grid refinement, similar to the one used in numerical schemes for the Bellman-equation (see [4]). For the adaptive time discretization we use a combination from step size con-

trol for ordinary differential equations and calculations for the rates of convergence of fully discrete solutions of the Bellman-equation (see [3]). We explain how both methods can be combined and applied to $Q$-learning. A simple numerical example shows the effects of suboptimal state space to time discretization ratio, and provides an insight in the problems of coupling both schemes.

## 2  Error estimation for adaptive choice of grid

We want to approximate the optimal value function $V : \Omega \to \mathbb{R}$ in a state space $\Omega \subset \mathbb{R}^d$ of the following problem: Minimize

$$J(x, u(.)) := \int_0^\infty e^{-\rho\tau} g(y_{x,u(.)}(\tau), u(\tau))d\tau, \quad u(.) : \mathbb{R}_+ \to A \text{ measurable}, \quad (1)$$

where $g : \Omega \times A \to \mathbb{R}_+$ is the cost function, and $y_{x,u(.)}(.)$ is the solution of the differential equation

$$\dot{y}(t) = f(y(t), u(t)), \quad y(0) = x. \quad (2)$$

As a trial space for the approximation of the optimal value function (or $Q$-function) we use locally linear elements on simplizes $S_i$, $i = 1, \ldots, N_S$ which form a triangulation of the state space, $N_S$ the number of simplizes. The vertices shall be called $x_i$, $i = 1, \ldots, N$, $N$ the dimension of the trial space[1]. This approach has been used in numerical schemes for the Bellman-equation ([2], [4]). We will first assume, that the grid is fixed and has a discretization parameter

$$k = \max_i \text{diam}\{S_i\}.$$

Other than in the numerical case, where the updates are performed in the vertices of the triangulation, in reinforcement learning only observed information is available. We will assume, that in one time step of size $h > 0$, we obtain the following information:

- the current state $y_n \in \Omega$,
- an action $a_n \in A$,
- the subsequent state $y_{n+1} := y_{y_n,a_n}(h)$
- the local cost $r_n = r(y_n, a_n) = \int_0^h e^{-\rho\tau} g(y_{y_n,a_n}(\tau), a_n(\tau))d\tau$ .

The state $y_n$, in which an update is to be made, may be any state in $\Omega$. $A$ shall be finite, and $a_n$ locally constant.

The new value of the fully discrete $Q$-function $Q_h^k(y_n, a_n)$ should be set to

$$Q_h^k(y_n, a_n) \quad \text{shall be} \quad r_n + e^{-\rho h} V_h^k(y_{n+1}),$$

where $V_h^k(y_{n+1}) = \min_a Q_h^k(y_{n+1}, a)$. We call the right side the update function

$$P_h(z, a, V_h^k) := r(z, a) + e^{-\rho h} V_h^k(y_{z,a}(h)), \quad z \in \Omega. \quad (3)$$

We will update $Q_h^k$ in the vertices $\{x_i\}_{i=1}^N$ of the triangulation in one of the following two ways:

**Kaczmarz-update.** Let $\lambda^T = (\lambda_1, \ldots, \lambda_N)$ be the vector of barycentric coordinates, such that

$$y_n = \sum_{i=1}^{N} \lambda_i x_i, \quad 0 \le \lambda_i \le 1, \text{ for all } i = 1, \ldots, N.$$

Then update

$$Q_h^k(x_i, a_n) := Q_h^k(x_i, a_n) + \frac{\lambda_i}{\lambda^T \lambda} \left[ r_n + e^{-\rho h} V_h^k(y_{n+1}) - \sum_{i=1}^{N} \lambda_i Q_h^k(x_i, a_n) \right]. \quad (4)$$

**Kronecker-update.** Let $S \ni y_n$ and $x$ be the vertex of $S$, closest to $y_n$ (if there is a draw, then the update can be performed in all winners). Then update $Q_h^k$ only in $x$ according to

$$Q_h^k(x, a_n) := r_n + e^{-\rho h} V_h^k(y_{n+1}). \quad (5)$$

Each method has some assets and drawbacks. In our computer simulations the Kaczmarz-update seemed to be more stable over the Kronecker-update (see [6]). However, examples may be constructed where a (Hölder-)continuous bounded optimal value function $V$ is to be approximated, and the Kaczmarz-update produces an approximation with arbitrarily high $\|.\|_{\sup}$-norm (place a vertex $x$ of the triangulation in a point where $\frac{d}{dx} V$ is infinity, and use as update states the vertex $x$ in turn with an arbitrarily close state $\bar{x}$).

Kronecker-update will provide a bounded approximation if $V$ is bounded. Let $\bar{V}_h^k$ be the fully-discrete optimal value function

$$\bar{V}_h^k(x_i) = \min_a \{ r(x_i, a) + e^{-\rho h} V_h^k(y_{x_i, a}(h)), \quad i = 1, \ldots, N.$$

Then it can be shown, that an approximation performed by Kronecker-update will eventually be caught in an $\varepsilon$-neighborhood of $\bar{V}_h^k$ (with respect to the $\|.\|_{\sup}$-norm), if the data points $y_0, y_1, y_2, \ldots$ are sufficiently dense. Under regularity conditions on $V$, $\varepsilon$ may be bounded by[2]

$$\varepsilon \le C\left(h + \frac{k}{h}\right). \quad (6)$$

As a criterion for grid refinement we choose a form of a *local a posteriori error estimate* as defined in [4]. Let $V_h^k(x) = \min_a Q_h^k(x, a)$ be the current iterate of the optimal value function. Let $a_x \in U$ be the minimizing control $a_x = \arg\min_a Q_h^k(x, a)$. Then we define

$$e_h(x) := |V_h^k(x) - P(x, a_x, V_h^k)|. \quad (7)$$

If $V_h^k$ is in the $\varepsilon$-neighborhood of $\bar{V}_h^k$, then it can be shown, that (for every $x \in \Omega$ and simplex $S_x$ with $x \in S_x$, $a_x$ as above)

$$0 \le e(x) \le \sup_{z \in S_x} P(z, a_z, V_h^k) - \inf_{z \in S_x} P(z, a_z, V_h^k).$$

If $\bar{V}_h^k$ is Lipschitz-continuous, then an estimate using only Gronwall's inequality bounds the right side and therefore $e(x)$ by $C\frac{k}{\rho h}$, where $C$ depends on the Lipschitz-constants of $\bar{V}_h^k$ and the cost $g$.

The value $e_j := \max_{x \in S_j} e_h(x)$ defines a function, which is locally constant on every simplex. We use $e_j$, $j = 1, \ldots, N$ as an indicator function for grid refinement. The (global) tolerance value $\text{tol}_k$ for $e_j$ shall be set to

$$\text{tol}_k = C * (\sum_{i=1}^{N_S} e_i)/N_S,$$

where we have chosen $1 \leq C \leq 2$. We approximate the function $e$ on the simplizes in the following way, starting in some $y_n \in S_j$:

---

1. apply a control $a \in U$ constantly on $[T, T + h]$
2. receive value $r_n$ and subsequent state $y_{n+1}$
3. calculate the update value $P_h(x, a, V_h^k)$
4. if $(|P_h(x, a, V_h^k) - V_h^k(x)| \geq e_j)$ then $e_j := |P_h(x, a, V_h^k) - V_h^k(x)|$

---

It is advisable to make grid refinements in one sweep. We also store (different to the described algorithm) several past values of $e_j$ in every simplex, to be able to distinguish between large $e_j$ due to few visits in that simplex and the large $e_j$ due to space discretization error. For grid refinement we use a method described in ([1]).

## 3   A local criterion for time refinement

*Why* not take the smallest possible sampling rate? There are two arguments for adaptive time discretization. First, a bigger time step $h$ naturally improves (decreases) the contraction rate of the iteration, which is $e^{-\rho h}$. The new information is conveyed from a point further away (in the future) for big $h$, without the need to store intermediate states along the trajectory. It is therefore reasonable to start with a big $h$ and refine where needed.

The second argument is, that the grid and time discretization $k$ and $h$ stand in a certain relation. In [3] the estimate

$$|V(x) - V_h^k(x)| \leq C(h + \frac{k}{\sqrt{h}}), \quad \text{for all } x \in \Omega, \quad C \text{ a constant}$$

is proven (or similar estimates, depending on the regularity of $V$). For obvious reasons, it is desirable to start with a coarse grid (storage, speed), i.e. $k$ large. Having a too small $h$ in this case will make the approximation error large. Also here, it is reasonable to start with a big $h$ and refine where needed.

*What* can serve as a refinement criterion for the time step $h$? In numerical schemes for ordinary differential equations, adaptive step size control is performed by estimating the local truncation error of the Taylor series by inserting intermediate points. In reinforcement learning, however, suppose the system has a large truncation error (i.e. it is difficult to control) in a certain region using large $h$ and locally constant control functions. If the optimal value function is nearly constant in this region, we will not have to refine $h$. The criterion must be, that at an intermediate point, e.g. at time $h/2$, the optimal value function assumes a value considerably smaller (better) than at time $h$. However, if this better value is due to error in the state discretization, then do not refine the time step.

We define a function $H$ on the simplices of the triangulation. $H(S) > 0$ holds the time-step which will be used when in simplex $S$. Starting at a state $y_n \in \Omega$, $y_n \in S_n$ at time $T > 0$, with the current iterate of the Q-function $Q_h^k$ ($V_h^k$ respectively) the following is performed:

> 1. apply a control $a \in U$ constantly on $[T, T+h]$
> 2. take a sample at the intermediate state $z = y_{y_n,a}(h/2)$
> 3. if $(H(S_n) < \texttt{C}*\sqrt{\text{diam}\{S_n\}})$ then end.
>    else:
> 4. compute $V_h^k(z) = \min_b Q_h^k(z, b)$
> 5. compute $P_{h/2}(y_n, a, V_h^k) = r_{h/2}(y_n, a) + e^{-\rho h/2} V_h^k(z)$
> 6. compute $P_h(y_n, a, V_h^k) = r_h(y_n, a) + e^{-\rho h} V_h^k(y_{n+1})$
> 7. if $(P_{h/2}(y_n, a, V_h^k) \leq P_h(y_n, a, V_h^k) - \texttt{tol})$ update $H(S_n) = H(S_n)/2$

The value C is currently set to

$$\texttt{C} = C(y_n, a) = \frac{2}{\rho}|r_{h/2}(y_n, a) - r_h(y_n, a)|,$$

whereby a local value of $\frac{M_f L_g h^2}{\rho}$ is approximated, $M_f(x) = \max_a |f(x, a)|$, $L_g$ an approximation of $|\nabla g(x, a)|$ (if $g$ is sufficiently regular).

tol depends on the local value of $V_h^k$ and is set to

$$\texttt{tol}(x) = 0.1 * V_h^k(x).$$

*How* can a $Q$-function $Q_{h(x)}^{k(x)}(x, a)$, with state dependent time and space discretisation be approximated and stored? We have stored the time discretisation function $H$ locally constant on every simplex. This implies (if $H$ is not constant on $\Omega$), that there will be vertices $x_j$, such that adjacent triangles hold different values of $H$. The $Q$-function, which is stored in the vertices, then has different choices of $H(x_j)$. We solved this problem, by updating a function $Q_H^k(x_j, a)$ with Kaczmarz-update and the update value $P_{H(y_n)}(y_n, a, V_h^k)$, $y_n$ in an to $x_j$ adjacent simplex, regardless of the different $H$-values in $x_j$. $Q_H^k(x_j, a)$ therefore has an ambiguous semantic: it is the value if $a$ is applied for 'some time', and optimal from there on. 'some time' depends here on the value of $H$ in the current simplex. It can be shown, that $|Q_{H(x_j)/2}^k(x_j, a) - Q_{H(x_j)}^k(x_j, a)|$ is less than the space discretization error.

## 4 A simple numerical example

We demonstrate the effects of suboptimal values for space and time discretisation with the following problem. Let the system equation be

$$\dot{y} = f(y, u) := \begin{pmatrix} u & 1 \\ -1 & u \end{pmatrix} (y - v), \quad v = \begin{pmatrix} .375 \\ .375 \end{pmatrix}, \quad y \in \Omega = [0, 1] \times [0, 1] \quad (8)$$

The stationary point of the uncontrolled system is $v$. The eigenvalues of the system are $\{u + i, u - i\}$, $u \in [-c, c]$. The system is reflected at the boundary.

The goal of the optimal control shall be steer the solution along a given trajectory in state space (see figure 1), minimizing the integral over the distance from the current state to the given trajectory. The reinforcement or cost function is therefore chosen to be

$$g(y) = \text{dist}(L, y)^{\frac{1}{4}}, \quad (9)$$

where $L$ denotes the set of points in the given trajectory. The cost functional takes the form

$$J_\rho(y, a(.)) = \int_0^\infty e^{-\rho\tau} g(y_{y,a}(\tau)) d\tau. \quad (10)$$

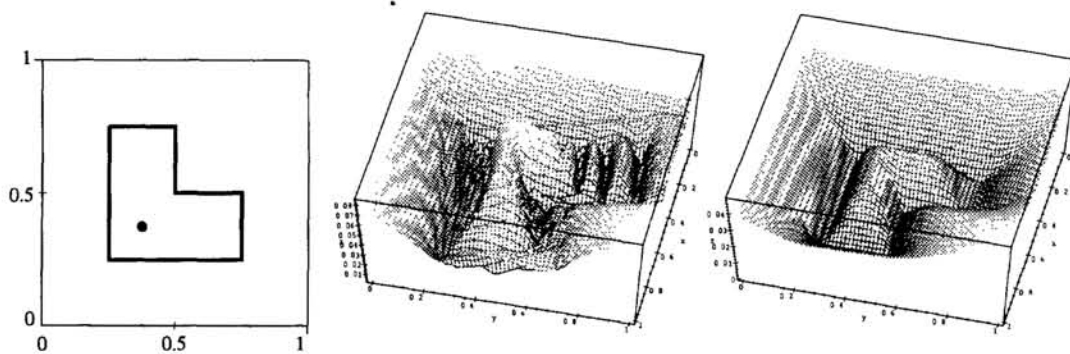

Figure 1: The left picture depicts the L-form of the given trajectory. The stationary point of the system is at (.375, .375) (depicted as a big dot). The optimal value function computed by numerical schemes on a fine fixed grid is depicted with too large time discretization (middle) and small time discretization (right) (rotated by about 100 degrees for better viewing). The waves in the middle picture show the effect of too large time steps in regions where $g$ varies considerably.

In the learning problem, the adaptive grid mechanism tries to resolve the waves (figure 1, middle picture) which come from the large time discretization. This is depicted in figure 2. We used only three different time step sizes ($h = 0.1$, 0.05 and 0.025) and started globally with the coarsest step size 0.1.

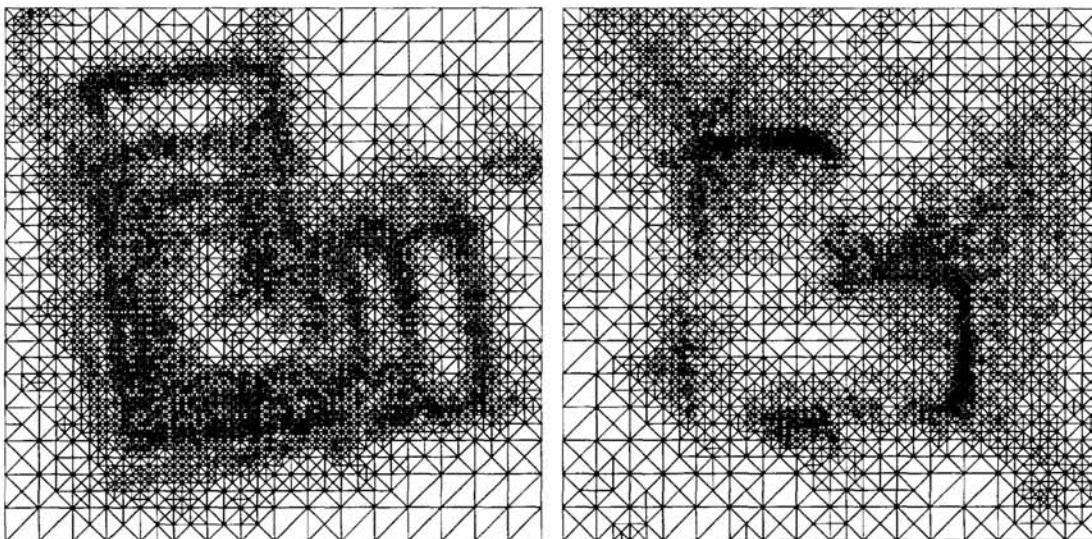

Figure 2: The adaptive grid mechanism refines correctly. However, in the left picture, unnecessary effort is spended in resolving regions, in which the time step should be refined urgently. The right picture shows the result, if adaptive time is also used. Regions outside the L-form are refined in the early stages of learning while $h$ was still large. An additional coarsening should be considered in future work. We used a high rate of random jumps in the process and locally a certainty equivalence controller to produce these pictures.

## 5   Discussion of the methods and conclusions

We described a time and space adaptive method for reinforcement learning with discounted cost functional. The ultimate goal would be, to find a self tuning algorithm which locally adjusted the time and space discretization automatically to the optimal ratio. The methods worked fine in the problems we investigated, e.g. nonlinearities in the system showed no problems. Nevertheless, the results depended on the choice of the tolerance values C, tol and $tol_k$. We used only three time discretization steps to prevent adjacent triangles holding time discretization values too far apart. The smallest state space resolution in the example is therefore too fine for the finest time resolution. A solution can be, to eventually use controls that are of higher order (in terms of approximation of control functions) than constant (e.g. linear, polynomial, or locally constant on subintervals of the finest time interval). This corresponds to locally open loop controls.

The optimality of the discretization ratio time/space could not be proven. Some discontinuous value functions $g$ gave problems, and we had problems handling stiff systems, too.

The learning period was considerably shorter (about factor 100 depending on the requested accuracy and initial data) in the adaptive cases as opposed to fixed grid and time with the same accuracy.

From our experience, it is difficult in numerical analysis to combine adaptive time and space discretization methods. To our knowledge this concept has not yet been applied to the Bellman-equation. Theoretical work is still to be done. We are aware, that triangulation of the state space yields difficulties in implementation in high dimensions. In future work we will be using rectangular grids. We will also make some comparisons with other algorithms like Parti-game ([5]). To us, a challenge is seen in handling discontinuous systems and cost functions as they appear in models with dry friction for example, as well as algebro-differential systems as they appear in robotics.

## Footnotes

[1]When an adaptive grid is used, then $N_S$ and $N$ depend on the refinement.

[2]With respect to the results in [3] we assume, that also $\varepsilon \le C(h + \frac{k}{\sqrt{h}})$ can be shown.

## References

[1]  E. Bänsch. Local mesh refinement in 2 and 3 dimensions. *IMPACT Comput. Sci. Engrg. 3, Vol. 3:181-191*, 1991.

[2]  M. Falcone. A numerical approach to the infinite horizon problem of deterministic control theory. *Appl Math Optim 15:1-13*, 1987.

[3]  R. Gonzalez and M. Tidball. On the rates of convergence of fully discrete solutions of Hamilton-Jacobi equations. *INRIA, Rapports de Recherche, No 1376, Programme 5*, 1991.

[4]  L. Grüne. An adaptive grid scheme for the discrete Hamilton-Jacobi-Bellman equation. *Numerische Mathematik, Vol. 75, No. 3:319-337*, 1997.

[5]  A. W. Moore and C. G. Atkeson. The parti-game algorithm for variable resolution reinforcement learning in multidimensional state-spaces. *Machine Learning, Volume 21*, 1995.

[6]  S. Pareigis. *Lernen der Lösung der Bellman-Gleichung durch Beobachtung von kontinuierlichen Prozeßen.* PhD thesis, Universität Kiel, 1996.

[7]  S. Pareigis. Multi-grid methods for reinforcement learning in controlled diffusion processes. In D. S. Touretzky, M. C. Mozer, and M. E. Hasselmo, editors, *Advances in Neural Information Processing Systems*, volume 9. The MIT Press, Cambridge, 1997.
